# Fast High-dimensional Kernel Summations Using the Monte Carlo Multipole Method

**Dongryeol Lee**
Computational Science and Engineering
Georgia Institute of Technology
Atlanta, GA 30332
dongryel@cc.gatech.edu

**Alexander Gray**
Computational Science and Engineering
Georgia Institute of Technology
Atlanta, GA 30332
agray@cc.gatech.edu

## Abstract

We propose a new fast Gaussian summation algorithm for high-dimensional datasets with high accuracy. First, we extend the original fast multipole-type methods to use approximation schemes with both hard and probabilistic error. Second, we utilize a new data structure called subspace tree which maps each data point in the node to its lower dimensional mapping as determined by any linear dimension reduction method such as PCA. This new data structure is suitable for reducing the cost of each pairwise distance computation, the most dominant cost in many kernel methods. Our algorithm guarantees probabilistic relative error on each kernel sum, and can be applied to high-dimensional Gaussian summations which are ubiquitous inside many kernel methods as the key computational bottleneck. We provide empirical speedup results on low to high-dimensional datasets up to 89 dimensions.

## 1 Fast Gaussian Kernel Summation

In this paper, we propose new computational techniques for efficiently approximating the following sum for each *query point* $q_i \in \mathcal{Q}$:

$$\Phi(q_i, \mathcal{R}) = \sum_{r_j \in \mathcal{R}} e^{-||q_i - r_j||^2/(2h^2)} \tag{1}$$

where $\mathcal{R}$ is the *reference set*; each *reference point* is associated with a Gaussian function with a smoothing parameter $h$ (the 'bandwidth'). This form of summation is ubiquitous in many statistical learning methods, including kernel density estimation, kernel regression, Gaussian process regression, radial basis function networks, spectral clustering, support vector machines, and kernel PCA [1, 4]. Cross-validation in all of these methods require evaluating Equation 1 for multiple values of $h$. Kernel density estimation, for example, requires $|\mathcal{R}|$ density estimate based on $|\mathcal{R}| - 1$ points, yielding a brute-force computational cost scaling **quadratically** (that is $O(|\mathcal{R}|^2)$).

**Error bounds.** Due to its expensive computational cost, many algorithms approximate the Gaussian kernel sums at the expense of reduced precision. Therefore, it is natural to discuss error bound criteria which measure the quality of the approximations with respect to their corresponding true values. The following error bound criteria are common in literature:

**Definition 1.1.** *An algorithm guarantees $\epsilon$ **absolute error bound**, if for each exact value $\Phi(q_i, \mathcal{R})$ for $q_i \in \mathcal{Q}$, it computes $\widetilde{\Phi}(q_i, \mathcal{R})$ such that $\left| \widetilde{\Phi}(q_i, \mathcal{R}) - \Phi(q_i, \mathcal{R}) \right| \leq \epsilon$.*

**Definition 1.2.** *An algorithm guarantees $\epsilon$ **relative error bound**, if for each exact value $\Phi(q_i, \mathcal{R})$ for $q_i \in \mathcal{Q}$, it computes $\widetilde{\Phi}(q_i, \mathcal{R}) \in \mathbb{R}$ such that $\left| \widetilde{\Phi}(q_i, \mathcal{R}) - \Phi(q_i, \mathcal{R}) \right| \leq \epsilon \left| \Phi(q_i, \mathcal{R}) \right|$.*

Bounding the relative error (e.g., the percentage deviation) is much harder because the error bound criterion is in terms of the initially unknown exact quantity. As a result, many previous methods [7] have focused on bounding the absolute error. The relative error bound criterion is preferred to the absolute error bound criterion in statistical applications in which high accuracy is desired. Our new algorithm will enforce the following "relaxed" form of the relative error bound criterion, whose motivation will be discussed shortly.

**Definition 1.3.** *An algorithm guarantees* $(1 - \alpha)$ **probabilistic** $\epsilon$ **relative error bound**, *if for each exact value* $\Phi(q_i, \mathcal{R})$ *for* $q_i \in \mathcal{Q}$, *it computes* $\widetilde{\Phi}(q_i, \mathcal{R}) \in \mathbb{R}$, *such that with at least probability* $0 < 1 - \alpha < 1$, $\left| \widetilde{\Phi}(q_i, \mathcal{R}) - \Phi(q_i, \mathcal{R}) \right| \leq \epsilon \left| \Phi(q_i, \mathcal{R}) \right|$.

**Previous work.** The most successful class of acceleration methods employ "higher-order divide and conquer" or *generalized N-body algorithms (GNA)* [4]. This approach can use any spatial partioning tree such as *kd*-trees or ball-trees for both the query set $\mathcal{Q}$ and reference data $\mathcal{R}$ and performs a simulataneous recursive descent on both trees.

*GNA* with relative error bounds (Definition 1.2) [5, 6, 11, 10] utilized bounding boxes and additional *cached-sufficient statistics* such as higher-order moments needed for series-expansion. [5, 6] utilized bounding-box based error bounds which tend to be very loose, which resulted in slow empirical performance around suboptimally small and large bandwidths. [11, 10] extended GNA-based Gaussian summations with series-expansion which provided tighter bounds; it showed enormous performance improvements, but only up to low dimensional settings (up to $D = 5$) since the number of required terms in series expansion increases exponentially with respect to $D$.

[9] introduces an iterative sampling based GNA for accelerating the computation of nested sums (a related easier problem). Its speedup is achieved by replacing pessimistic error bounds provided by bounding boxes with normal-based confidence interval from Monte Carlo sampling. [9] demonstrates the speedup many orders of magnitude faster than the previous state of the art in the context of computing aggregates over the queries (such as the LSCV score for selecting the optimal bandwidth). However, the authors did not discuss the sampling-based approach for computations that require *per-query* estimates, such as those required for kernel density estimation.

None of the previous approaches for kernel summations addresses the issue of reducing the computational cost of each distance computation which incurs $O(D)$ cost. However, the *intrinsic dimensionality* $d$ of most high-dimensional datasets is much smaller than the explicit dimension $D$ (that is, $d << D$). [12] proposed tree structures using a global dimension reduction method, such as random projection, as a preprocessing step for efficient $(1 + \epsilon)$ approximate nearest neighbor search. Similarly, we develop a new data structure for kernel summations; our new data structure is constructed in a top-down fashion to perform the initial spatial partitioning in the original input space $\mathbb{R}^D$ and performs a local dimension reduction to a localized subset of the data in a bottom-up fashion.

**This paper.** We propose a new fast Gaussian summation algorithm that enables speedup in higher dimensions. Our approach utilizes: 1) probabilistic relative error bounds (Definition 1.3) on kernel sums provided by Monte Carlo estimates 2) a new tree structure called *subspace tree* for reducing the computational cost of each distance computation. The former can be seen as relaxing the strict requirement of guaranteeing hard relative bound on very small quantities, as done in [5, 6, 11, 10]. The latter was mentioned as a possible way of ameliorating the effects of *the curse of dimensionality* in [14], a pioneering paper in this area.

**Notations.** Each *query point* and *reference point* (a $D$-dimensional vector) is indexed by natural numbers $i, j \in \mathbb{N}$, and denoted $q_i$ and $r_j$ respectively. For any set $S$, $|S|$ denotes the number of elements in $S$. The entities related to the left and the right child are denoted with superscripts $L$ and $R$; an internal node $N$ has the child nodes $N^L$ and $N^R$.

## 2    Gaussian Summation by Monte Carlo Sampling

Here we describe the extension needed for probabilistic computation of kernel summation satisfying Definition 1.3. The main routine for the probabilistic kernel summation is shown in Algorithm 1. The function MCMM takes the *query node* $Q$ and the *reference node* $R$ (each initially called with the roots of the *query tree* and the *reference tree*, $Q^{root}$ and $R^{root}$) and $\beta$ (initially called with $\alpha$ value which controls the probability guarantee that each kernel sum is within $\epsilon$ relative error).

---
**Algorithm 1** The core dual-tree routine for probabilistic Gaussian kernel summation.
---
   MCMM$(Q, R, \beta)$
     **if** CANSUMMARIZEEXACT$(Q, R, \epsilon)$ **then**
       SUMMARIZEEXACT$(Q, R)$
     **else if** CANSUMMARIZEMC$(Q, R, \epsilon, \beta)$ **then**
5:     SUMMARIZEMC$(Q, R, \epsilon, \beta)$
     **else**
       **if** $Q$ is a leaf node **then**
         **if** $R$ is a leaf node **then**
           MCMMBASE$(Q, R)$
10:       **else**
           MCMM$\left(Q, R^L, \frac{\beta}{2}\right)$, MCMM$\left(Q, R^R, \frac{\beta}{2}\right)$
       **else**
         **if** $R$ is a leaf node **then**
           MCMM$(Q^L, R, \beta)$, MCMM$(Q^R, R, \beta)$
15:       **else**
           MCMM$\left(Q^L, R^L, \frac{\beta}{2}\right)$, MCMM$\left(Q^L, R^R, \frac{\beta}{2}\right)$
           MCMM$\left(Q^R, R^L, \frac{\beta}{2}\right)$, MCMM$\left(Q^R, R^R, \frac{\beta}{2}\right)$
---

The idea of Monte Carlo sampling used in the new algorithm is similar to the one in [9], except the sampling is done per query and we use approximations that provide hard error bounds as well (i.e. finite difference, exhaustive base case: MCMMBASE). This means that the approximation has less variance than a pure Monte Carlo approach used in [9]. Algorithm 1 first attempts approximations with hard error bounds, which are computationally cheaper than sampling-based approximations. For example, finite-difference scheme [5, 6] can be used for the CANSUMMARIZEEXACT and SUMMARIZEEXACT functions in any general dimension.

The CANSUMMARIZEMC function takes two parameters that specify the accuracy: the relative error and its probability guarantee and decides whether to use Monte Carlo sampling for the given pair of nodes. If the *reference node* $R$ contains too few points, it may be more efficient to process it using exact methods that use error bounds based on bounding primitives on the node pair or exhaustive pair-wise evaluations, which is determined by the condition: $\tau \cdot m_{initial} \leq |R|$ where $\tau > 1$ controls the minimum number of *reference points* needed for Monte Carlo sampling to proceed. If the *reference node* does contain enough points, then for each *query point* $q \in Q$, the SAMPLE routine samples $m_{initial}$ terms over the terms in the summation $\Phi(q, R) = \sum\limits_{r_{j_n} \in R} K_h(||q - r_{j_n}||)$

where $\Phi(q, R)$ denotes the exact contribution of $R$ to $q$'s kernel sum. Basically, we are interested in estimating $\Phi(q, R)$ by $\widetilde{\Phi}(q, R) = |R|\mu_S$, where $\mu_S$ is the sample mean of $S$. From the Central Limit Theorem, given enough $m$ samples, $\mu_S \leadsto N(\mu, \sigma_S^2/m)$ where $\Phi(q, R) = |R|\mu$ (i.e. $\mu$ is the average of the kernel value between $q$ and any *reference point* $r \in R$); this implies that $|\mu_S - \mu| \leq z_{\beta/2}\sigma_S/\sqrt{m}$ with probability $1 - \beta$. The *pruning rule* we have to enforce for each *query point* for the contribution of $R$ is:

$$z_{\beta/2}\frac{\sigma_S}{\sqrt{m}} \leq \frac{\epsilon\Phi(q, \mathcal{R})}{|\mathcal{R}|}$$

where $\sigma_S$ the sample standard deviation of $S$. Since $\Phi(q, \mathcal{R})$ is one of the unknown quanities we want to compute, we instead enforce the following:

$$z_{\beta/2}\frac{\sigma_S}{\sqrt{m}} \leq \frac{\epsilon\left(\Phi^l(q, \mathcal{R}) + |R|\left(\mu_S - \frac{z_{\beta/2}\sigma_S}{\sqrt{m}}\right)\right)}{|\mathcal{R}|} \qquad (2)$$

where $\Phi^l(q, \mathcal{R})$ is the currently running lower bound on the sum computed using exact methods and $|R|\left(\mu_S - \frac{z_{\beta/2}\sigma_S}{\sqrt{m}}\right)$ is the probabilistic component contributed by $R$. Denoting $\Phi^{l,new}(q, \mathcal{R}) = \Phi^l(q, \mathcal{R}) + |R|\left(\mu_S - \frac{z_{\beta/2}\sigma_S}{\sqrt{|S|}}\right)$, the **minimum number of samples for** $q$ needed to achieve the

target error the right side of the inequality in Equation 2 with at least probability of $1 - \beta$ is:

$$m \geq z_{\beta/2}^2 \sigma_S^2 \frac{(|\mathcal{R}| + \epsilon|R|)^2}{\epsilon^2(\Phi^l(q, \mathcal{R}) + |R|\mu_S)^2}$$

If the given *query node* and *reference node* pair cannot be pruned using either non-probabilistic/probabilistic approximations, then we recurse on a smaller subsets of two sets. In particular, when dividing over the *reference node* $R$, we recurse with half of the $\beta$ value[1]. We now state the probablistic error guarantee of our algorithm as a theorem.

**Theorem 2.1.** *After calling* MCMM *with* $Q = Q^{root}$, $R = R^{root}$, *and* $\beta = \alpha$, *Algorithm 1 approximates each* $\Phi(q, \mathcal{R})$ *with* $\widetilde{\Phi}(q, \mathcal{R})$ *such that Definition 1.3 holds.*

*Proof.* For a *query/reference* $(Q, R)$ pair and $0 < \beta < 1$, MCMMBASE and SUMMARIZEEXACT compute estimates for $q \in Q$ such that $\left|\widetilde{\Phi}(q, R) - \Phi(q, R)\right| < \epsilon\frac{\Phi(q,\mathcal{R})|R|}{|\mathcal{R}|}$ with probability at least $1 > 1 - \beta$. By Equation 2, SUMMARIZEMC computes estimates for $q \in Q$ such that $\left|\widetilde{\Phi}(q, R) - \Phi(q, R)\right| < \epsilon\frac{\Phi(q,\mathcal{R})|R|}{|\mathcal{R}|}$ with probability $1 - \beta$.

We now induct on $|Q \cup R|$. Line 11 of Algorithm 1 divides over the *reference* whose subcalls compute estimates that satisfy $\left|\widetilde{\Phi}(q, R^L) - \Phi(q, R^L)\right| \leq \epsilon\frac{\Phi(q,R)|R^L|}{|\mathcal{R}|}$ and $\left|\widetilde{\Phi}(q, R^R) - \Phi(q, R^R)\right| \leq \epsilon\frac{\Phi(q,R)|R^R|}{|\mathcal{R}|}$ each with at least $1 - \frac{\beta}{2}$ probability by induction hypothesis. For $q \in Q$, $\widetilde{\Phi}(q, R) = \widetilde{\Phi}(q, R^L) + \widetilde{\Phi}(q, R^R)$ which means $|\widetilde{\Phi}(q, R) - \Phi(q, R)| \leq \epsilon\frac{\Phi(q,\mathcal{R})|R|}{|\mathcal{R}|}$ with probability at least $1 - \beta$. Line 14 divides over the *query* and each subcall computes estimates that hold with at least probability $1 - \beta$ for $q \in Q^L$ and $q \in Q^R$. Line 16 and 17 divides both over the *query* and the *reference*, and the correctness can be proven similarly. Therefore, $MCMM(Q^{root}, R^{root}, \alpha)$ computes estimates satisfying Definition 1.3. □

**"Reclaiming" probability.** We note that the assigned probability $\beta$ for the *query/reference* pair computed with exact bounds (SUMMARIZEEXACT and MCMMBASE) is not used. This portion of the probability can be "reclaimed" in a similar fashion as done in [10] and re-used to prune more aggressively in the later stages of the algorithm. All experiments presented in this paper were benefited by this simple modification.

## 3 Subspace Tree

A subspace tree is basically a space-partitioning tree with a set of orthogonal bases associated with each node $N$: $N.\Omega = (\mu, U, \Lambda, d)$ where $\mu$ is the mean, $U$ is a $D \times d$ matrix whose columns consist of $d$ eigenvectors, and $\Lambda$ the corresponding eigenvalues. The orthogonal basis set is constructed using a linear dimension reduction method such as PCA. It is constructed in the top-down manner using the PARTITIONSET function dividing the given set of points into two (where the PARTITIONSET function divides along the dimension with the highest variance in case of a *kd*-tree for example), with the subspace in each node formed in the bottom-up manner. Algorithm 3 shows a PCA tree (a subspace tree using PCA as a dimension reduction) for a 3-D dataset. The subspace of each leaf node is computed using PCABASE which can use the exact PCA [3] or a stochastic one [2]. For an internal node, the subspaces of the child nodes, $N^L.\Omega = (\mu^L, U^L, \Lambda^L, d^L)$ and $N^R.\Omega = (\mu^R, U^R, \Lambda^R, d^R)$, are approximately merged using the MERGESUBSPACES function which involves solving an $(d^L + d^R + 1) \times (d^L + d^R + 1)$ eigenvalue problem [8], which runs in $O((d^L + d^R + 1)^3) << O(D^3)$ given that the dataset is sparse. In addition, each data point $x$ in each node $N$ is mapped to its new lower-dimensional coordinate using the orthogonal basis set of $N$: $x_{proj} = U^T(x - \mu)$. The $L_2$ norm reconstruction error is given by: $||x_{recon} - x||_2^2 = ||(Ux_{proj} + \mu) - x||_2^2$.

**Monte Carlo sampling using a subspace tree.** Consider CANSUMMARIZEMC function in Algorithm 2. The "outer-loop" over this algorithm is over the *query set* $Q$, and it would make sense to project each *query point* $q \in Q$ to the subspace owned by the *reference node* $R$. Let $U$ and $\mu$ be the orthogonal basis system for $R$ consisting of $d$ basis. For each $q \in Q$, consider the squared distance

**Algorithm 2** Monte Carlo sampling based approximation routines.

| SAMPLE$(q, R, \epsilon, \alpha, S, m)$ | CANSUMMARIZEMC$(Q, R, \epsilon, \alpha)$ |
|---|---|
| **for** $k = 1$ to $m$ **do** | **return** $\tau \cdot m_{initial} \leq \|R\|$ |
| $\quad r \leftarrow$ random point in $R$ | |
| $\quad S \leftarrow S \cup \{K_h(\|q - r\|)\}$ | SUMMARIZEMC$(Q, R, \epsilon, \alpha)$ |
| $\mu_S \leftarrow$ MEAN$(S), \quad \sigma_S^2 \leftarrow$ VARIANCE$(S)$ | **for** $q_i \in Q$ **do** |
| $\Phi^{l,new}(q, \mathcal{R}) \leftarrow \Phi^l(q, \mathcal{R}) + \|R\| \left( \mu_S - \frac{z_{\alpha/2}\sigma_S}{\sqrt{\|S\|}} \right)$ | $\quad S \leftarrow \emptyset, m \leftarrow m_{initial}$ |
| | $\quad$ **repeat** |
| $m_{thresh} \leftarrow z_{\alpha/2}^2 \sigma_S^2 \frac{(\|\mathcal{R}\| + \epsilon\|R\|)^2}{\epsilon^2 (\Phi^l(q,\mathcal{R}) + \|R\|\mu_S)^2}$ | $\quad\quad$ SAMPLE$(q_i, R, \epsilon, \alpha, S, m)$ |
| $m \leftarrow m_{thresh} - \|S\|$ | $\quad$ **until** $m \leq 0$ |
| | $\quad \Phi(q_i, \mathcal{R}) \leftarrow \Phi(q_i, \mathcal{R}) + \|R\| \cdot$ MEAN$(S)$ |

$\|(q - \mu) - r_{proj}\|^2$ (where $(q - \mu)$ is $q$'s coordinates expressed in terms of the coordinate system of $R$) as shown in Figure 1. For the Gaussian kernel, each pairwise kernel value is approximated as:

$$e^{-\|q-r\|^2/(2h^2)} \approx e^{-\|q-q_{recon}\|^2/(2h^2)} e^{-\|q_{proj} - r_{proj}\|^2/(2h^2)} \tag{3}$$

where $q_{recon} = U q_{proj} + \mu$ and $q_{proj} = U^T(q - \mu)$. For a fixed *query point* $q$, $e^{-\|q-q_{recon}\|^2/(2h^2)}$ can be precomputed (which takes $d$ dot products between two $D$-dimensional vectors) and re-used for every distance computation between $q$ and any *reference point* $r \in R$ whose cost is now $O(d) << O(D)$. Therefore, we can take more samples efficiently. For a total of sufficiently large $m$ samples, the computational cost is $O(d(D + m)) << O(D \cdot m)$ for each *query point*.

Increased variance comes at the cost of inexact distance computations, however. Each distance computation incurs at most squared $L_2$ norm of $\|r_{recon} - r\|_2^2$ error. That is, $\left| \|q - r_{recon}\|_2^2 - \|q - r\|_2^2 \right| \leq \|r_{recon} - r\|_2^2$. Nevertheless, the sample variance for each *query point* **plus** the inexactness due to dimension reduction $\tau_S$ can be shown to be bounded for the Gaussian kernel as: (where each $s = e^{-\|q - r_{recon}\|^2/(2h^2)}$):

$$\frac{1}{m-1} \left( \sum_{s \in S} s^2 - m \cdot \mu_S^2 \right) + \tau_S$$

$$\leq \frac{1}{m-1} \left( \left( \sum_{s \in S} s^2 \right) \min \left\{ 1, \max_{r \in R} e^{\|r_{recon} - r\|_2^2/h^2} \right\} - m \left( \mu_S \min_{r \in R} e^{-\|r_{recon} - r\|_2^2/(2h^2)} \right)^2 \right)$$

**Exhaustive computations using a subspace tree.** Now suppose we have built subspace trees for the *query* and the *reference* sets. We can project either each *query point* onto the *reference* subspace, or each *reference point* onto the *query* subspace, depending on which subspace has a smaller dimension and the number of points in each node. The subspaces formed in the leaf nodes usually are highly numerically accurate since it contains very few points compared to the extrinsic dimensionality $D$.

## 4 Experimental Results

We empirically evaluated the runtime performance of our algorithm on seven real-world datasets, scaled to fit in $[0,1]^D$ hypercube, for approximating the Gaussian sum at every query point with a range of bandwidths. This experiment is motivated by many kernel methods that require computing the Gaussian sum at different bandwidth values (according to the standard least-sqares cross-validation scores [15]). Nevertheless, we emphasize that the acceleration results are applicable to other kernel methods that require efficient Gaussian summation.

In this paper, the *reference set* equals the *query set*. All datasets have 50K points so that the exact exhaustive method can be tractably computed. All times are in seconds and include the time needed to build the trees. Codes are in C/C++ and run on a dual Intel Xeon 3GHz with 8 Gb of main memory. The measurements in second to eigth columns are obtained by running the algorithms at the bandwidth $kh^*$ where $10^{-3} \leq k \leq 10^3$ is the constant in the corresponding column header. The last columns denote the total time needed to run on all seven bandwidth values.

Each table has results for five algorithms: the naive algorithm and four algorithms. The algorithms with $p = 1$ denote the previous state-of-the-art (finite-difference with error redistribution) [10],

**Algorithm 3** PCA tree building routine.

---

BuildPcaTree($\mathcal{P}$)
  **if** CanPartition($\mathcal{P}$) **then**
    $\{\mathcal{P}^L, \mathcal{P}^R\} \leftarrow$ PartitionSet($\mathcal{P}$)
    $N \leftarrow$ empty node
    $N^L \leftarrow$ BuildPcaTree($\mathcal{P}^L$)
    $N^R \leftarrow$ BuildPcaTree($\mathcal{P}^R$)
    $N.S \leftarrow$ MergeSubspaces($N^L.S, N^R.S$)
  **else**
    $N \leftarrow$ BuildPcaTreeBase($\mathcal{P}$)
    $N.S \leftarrow$ PcaBase($\mathcal{P}$)
  $N.\mathcal{P}_{proj} \leftarrow$ Project($\mathcal{P}, N.S$)
  **return** $N$

---

while those with $p < 1$ denote our probabilistic version. Each entry has the running time and the percentage of the *query points* that **did not** satisfy the relative error $\epsilon$.

**Analysis.** Readers should focus on the last columns containing the total time needed for evaluating Gaussian sum at all points for seven different bandwidth values. This is indicated by boldfaced numbers for our probabilistic algorithm. As expected, On low-dimensional datasets (below 6 dimensions), the algorithm using series-expansion based bounds gives two to three times speedup compared to our approach that uses Monte Carlo sampling. Multipole moments are an effective form of compression in low dimensions with analytical error bounds that can be evaluated; our Monte Carlo-based method has an asymptotic error bound which must be "learned" through sampling.

As we go from 7 dimensions and beyond, series-expansion cannot be done efficiently because of its slow convergence. Our probabilistic algorithm ($p = 0.9$) using Monte Carlo consistently performs better than the algorithm using exact bounds ($p = 1$) by at least a factor of two. Compared to naive, it achieves the maximum speedup of about nine times on an 16-dimensional dataset; on an 89-dimensional dataset, it is at least three times as fast as the naive. Note that all the datasets contain only 50K points, and the speedup will be more dramatic as we increase the number of points.

## 5 Conclusion

We presented an extension to fast multipole methods to use approximation methods with both hard and probabilistic bounds. Our experimental results show speedup over the previous state-of-the-art on high-dimensional datasets. Our future work will include possible improvements inspired by a recent work done in the FMM community using a matrix-factorization formulation [13].

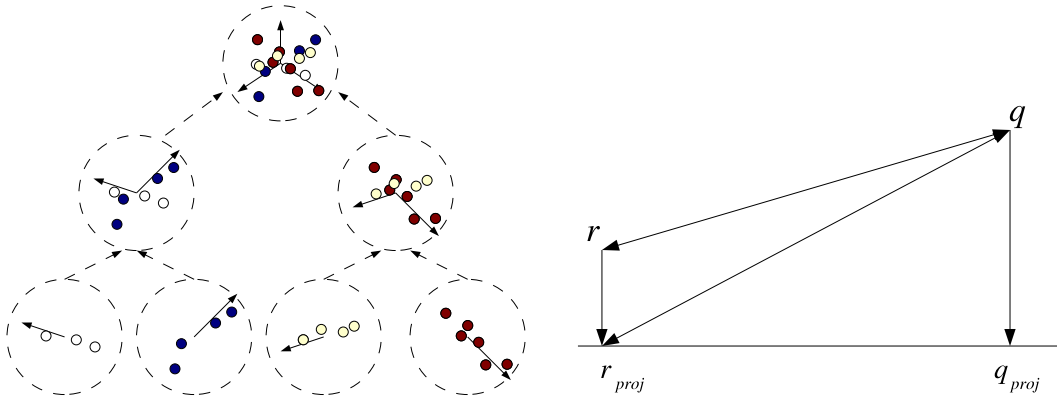

Figure 1: Left: A PCA-tree for a 3-D dataset. Right: The squared Euclidean distance between a given *query point* and a *reference point* projected onto a subspace can be decomposed into two components: the orthogonal component and the component in the subspace.

| Algorithm \ scale | 0.001 | 0.01 | 0.1 | 1 | 10 | 100 | 1000 | Σ |
|---|---|---|---|---|---|---|---|---|
| mockgalaxy-D-1M-rnd (cosmology: positions), $D=3, N=50000, h^*=0.000768201$ | | | | | | | | |
| *Naive* | 182 | 182 | 182 | 182 | 182 | 182 | 182 | 1274 |
| *MCMM* | 3 | 3 | 5 | 10 | 26 | 48 | 2 | **97** |
| $(\epsilon=0.1, p=0.9)$ | 1 % | 1 % | 1 % | 1 % | 1 % | 1 % | 5 % | |
| *DFGT* | 2 | 2 | 2 | 2 | 6 | 19 | 3 | 36 |
| $(\epsilon=0.1, p=1)$ | 0 % | 0 % | 0 % | 0 % | 0 % | 0 % | 0 % | |
| *MCMM* | 3 | 3 | 4 | 11 | 27 | 58 | 21 | **127** |
| $(\epsilon=0.01, p=0.9)$ | 0 % | 0 % | 1 % | 1 % | 1 % | 1 % | 7 % | |
| *DFGT* | 2 | 2 | 2 | 2 | 7 | 30 | 5 | 50 |
| $(\epsilon=0.01, p=1)$ | 0 % | 0 % | 0 % | 0 % | 0 % | 0 % | 0 % | |

| Algorithm \ scale | 0.001 | 0.01 | 0.1 | 1 | 10 | 100 | 1000 | Σ |
|---|---|---|---|---|---|---|---|---|
| bio5-rnd (biology: drug activity), $D=5, N=50000, h^*=0.000567161$ | | | | | | | | |
| *Naive* | 214 | 214 | 214 | 214 | 214 | 214 | 214 | 1498 |
| *MCMM* | 4 | 4 | 6 | 144 | 149 | 65 | 1 | **373** |
| $(\epsilon=0.1, p=0.9)$ | 0 % | 0 % | 0 % | 0 % | 1 % | 0 % | 1 % | |
| *DFGT* | 4 | 4 | 5 | 24 | 96 | 65 | 2 | 200 |
| $(\epsilon=0.1, p=1)$ | 0 % | 0 % | 0 % | 0 % | 0 % | 0 % | 0 % | |
| *MCMM* | 4 | 4 | 6 | 148 | 165 | 126 | 1 | **454** |
| $(\epsilon=0.01, p=0.9)$ | 0 % | 0 % | 0 % | 0 % | 1 % | 0 % | 1 % | |
| *DFGT* | 4 | 4 | 5 | 25 | 139 | 126 | 4 | 307 |
| $(\epsilon=0.01, p=1)$ | 0 % | 0 % | 0 % | 0 % | 0 % | 0 % | 0 % | |

| Algorithm \ scale | 0.001 | 0.01 | 0.1 | 1 | 10 | 100 | 1000 | Σ |
|---|---|---|---|---|---|---|---|---|
| $pall7-rnd, D=7, N=50000, h^*=0.00131865$ | | | | | | | | |
| *Naive* | 327 | 327 | 327 | 327 | 327 | 327 | 327 | 2289 |
| *MCMM* | 3 | 3 | 3 | 3 | 63 | 224 | < 1 | **300** |
| $(\epsilon=0.1, p=0.9)$ | 0 % | 0 % | 0 % | 1 % | 1 % | 12 % | 0 % | |
| *DFGT* | 10 | 10 | 11 | 14 | 84 | 263 | 223 | 615 |
| $(\epsilon=0.1, p=1)$ | 0 % | 0 % | 0 % | 0 % | 0 % | 0 % | 0 % | |
| *MCMM* | 3 | 3 | 3 | 3 | 70 | 265 | 5 | **352** |
| $(\epsilon=0.01, p=0.9)$ | 0 % | 0 % | 0 % | 1 % | 2 % | 1 % | 8 % | |
| *DFGT* | 10 | 10 | 11 | 14 | 85 | 299 | 374 | 803 |
| $(\epsilon=0.01, p=1)$ | 0 % | 0 % | 0 % | 0 % | 0 % | 0 % | 0 % | |

| Algorithm \ scale | 0.001 | 0.01 | 0.1 | 1 | 10 | 100 | 1000 | Σ |
|---|---|---|---|---|---|---|---|---|
| $covtype-rnd, D=10, N=50000, h^*=0.0154758$ | | | | | | | | |
| *Naive* | 380 | 380 | 380 | 380 | 380 | 380 | 380 | 2660 |
| *MCMM* | 11 | 11 | 13 | 39 | 318 | < 1 | < 1 | **381** |
| $(\epsilon=0.1, p=0.9)$ | 0 % | 0 % | 0 % | 1 % | 0 % | 0 % | 0 % | |
| *DFGT* | 26 | 27 | 38 | 177 | 390 | 244 | < 1 | 903 |
| $(\epsilon=0.1, p=1)$ | 0 % | 0 % | 0 % | 0 % | 0 % | 0 % | 0 % | |
| *MCMM* | 11 | 11 | 13 | 77 | 362 | 2 | < 1 | **477** |
| $(\epsilon=0.01, p=0.9)$ | 0 % | 0 % | 0 % | 1 % | 1 % | 10 % | 0 % | |
| *DFGT* | 26 | 27 | 38 | 180 | 427 | 416 | < 1 | 1115 |
| $(\epsilon=0.01, p=1)$ | 0 % | 0 % | 0 % | 0 % | 0 % | 0 % | 0 % | |

| Algorithm \ scale | 0.001 | 0.01 | 0.1 | 1 | 10 | 100 | 1000 | Σ |
|---|---|---|---|---|---|---|---|---|
| $CoocTexture-rnd, D=16, N=50000, h^*=0.0263958$ | | | | | | | | |
| *Naive* | 472 | 472 | 472 | 472 | 472 | 472 | 472 | 3304 |
| *MCMM* | 10 | 11 | 22 | 189 | 109 | < 1 | < 1 | **343** |
| $(\epsilon=0.1, p=0.9)$ | 0 % | 0 % | 0 % | 1 % | 8 % | 0 % | 0 % | |
| *DFGT* | 22 | 26 | 82 | 240 | 452 | 66 | < 1 | 889 |
| $(\epsilon=0.1, p=1)$ | 0 % | 0 % | 0 % | 0 % | 0 % | 0 % | 0 % | |
| *MCMM* | 10 | 11 | 22 | 204 | 285 | < 1 | < 1 | **534** |
| $(\epsilon=0.01, p=0.9)$ | 0 % | 0 % | 1 % | 1 % | 10 % | 4 % | 0 % | |
| *DFGT* | 22 | 26 | 83 | 254 | 543 | 230 | < 1 | 1159 |
| $(\epsilon=0.01, p=1)$ | 0 % | 0 % | 0 % | 0 % | 0 % | 0 % | 0 % | |

| Algorithm \ scale | 0.001 | 0.01 | 0.1 | 1 | 10 | 100 | 1000 | Σ |
|---|---|---|---|---|---|---|---|---|
| $LayoutHistogram - rnd, D = 32, N = 50000, h^* = 0.0609892$ | | | | | | | | |
| Naive | 757 | 757 | 757 | 757 | 757 | 757 | 757 | 5299 |
| MCMM | 32 | 32 | 54 | 168 | 583 | 8 | 8 | **885** |
| ($\epsilon = 0.1, p = 0.9$) | 0 % | 0 % | 1 % | 1 % | 1 % | 0 % | 0 % | |
| DFGT | 153 | 159 | 221 | 492 | 849 | 212 | < 1 | 2087 |
| ($\epsilon = 0.1, p = 1$) | 0 % | 0 % | 0 % | 0 % | 0 % | 0 % | 0 % | |
| MCMM | 32 | 45 | 60 | 183 | 858 | 8 | 8 | **1246** |
| ($\epsilon = 0.01, p = 0.9$) | 0 % | 0 % | 1 % | 6 % | 1 % | 0 % | 0 % | |
| DFGT | 153 | 159 | 222 | 503 | 888 | 659 | < 1 | 2585 |
| ($\epsilon = 0.01, p = 1$) | 0 % | 0 % | 0 % | 0 % | 0 % | 0 % | 0 % | |
| Algorithm \ scale | 0.001 | 0.01 | 0.1 | 1 | 10 | 100 | 1000 | Σ |
| $CorelCombined - rnd, D = 89, N = 50000, h^* = 0.0512583$ | | | | | | | | |
| Naive | 1716 | 1716 | 1716 | 1716 | 1716 | 1716 | 1716 | 12012 |
| MCMM | 384 | 418 | 575 | 428 | 1679 | 17 | 17 | **3518** |
| ($\epsilon = 0.1, p = 0.9$) | 0 % | 0 % | 0 % | 1 % | 10 % | 0 % | 0 % | |
| DFGT | 659 | 677 | 864 | 1397 | 1772 | 836 | 17 | 6205 |
| ($\epsilon = 0.1, p = 1$) | 0 % | 0 % | 0 % | 0 % | 0 % | 0 % | 0 % | |
| MCMM | 401 | 419 | 575 | 437 | 1905 | 17 | 17 | **3771** |
| ($\epsilon = 0.01, p = 0.9$) | 0 % | 0 % | 0 % | 1 % | 2 % | 0 % | 0 % | |
| DFGT | 659 | 677 | 865 | 1425 | 1794 | 1649 | 17 | 7086 |
| ($\epsilon = 0.01, p = 1$) | 0 % | 0 % | 0 % | 0 % | 0 % | 0 % | 0 % | |

## Footnotes

[1]We could also divide $\beta$ such that the node that may be harder to approximate gets a lower value.

## References

[1] Nando de Freitas, Yang Wang, Maryam Mahdaviani, and Dustin Lang. Fast krylov methods for n-body learning. In Y. Weiss, B. Schölkopf, and J. Platt, editors, *Advances in Neural Information Processing Systems 18*, pages 251–258. MIT Press, Cambridge, MA, 2006.

[2] P. Drineas, R. Kannan, and M. Mahoney. Fast monte carlo algorithms for matrices iii: Computing a compressed approximate matrix decomposition, 2004.

[3] G. Golub. *Matrix Computations, Third Edition*. The Johns Hopkins University Press, 1996.

[4] A. Gray and A. W. Moore. N-Body Problems in Statistical Learning. In Todd K. Leen, Thomas G. Dietterich, and Volker Tresp, editors, *Advances in Neural Information Processing Systems 13 (December 2000)*. MIT Press, 2001.

[5] Alexander G. Gray and Andrew W. Moore. Nonparametric Density Estimation: Toward Computational Tractability. In *SIAM International Conference on Data Mining 2003*, 2003.

[6] Alexander G. Gray and Andrew W. Moore. Very Fast Multivariate Kernel Density Estimation via Computational Geometry. In *Joint Statistical Meeting 2003*, 2003. to be submitted to JASA.

[7] L. Greengard and J. Strain. The Fast Gauss Transform. *SIAM Journal of Scientific and Statistical Computing*, 12(1):79–94, 1991.

[8] Peter Hall, David Marshall, and Ralph Martin. Merging and splitting eigenspace models. *IEEE Transactions on Pattern Analysis and Machine Intelligence*, 22(9):1042–1049, 2000.

[9] Michael Holmes, Alexander Gray, and Charles Isbell. Ultrafast monte carlo for statistical summations. In J.C. Platt, D. Koller, Y. Singer, and S. Roweis, editors, *Advances in Neural Information Processing Systems 20*, pages 673–680. MIT Press, Cambridge, MA, 2008.

[10] Dongryeol Lee and Alexander Gray. Faster gaussian summation: Theory and experiment. In *Proceedings of the Twenty-second Conference on Uncertainty in Artificial Intelligence*. 2006.

[11] Dongryeol Lee, Alexander Gray, and Andrew Moore. Dual-tree fast gauss transforms. In Y. Weiss, B. Schölkopf, and J. Platt, editors, *Advances in Neural Information Processing Systems 18*, pages 747–754. MIT Press, Cambridge, MA, 2006.

[12] Ting Liu, Andrew W. Moore, and Alexander Gray. Efficient exact k-nn and nonparametric classification in high dimensions. In Sebastian Thrun, Lawrence Saul, and Bernhard Schölkopf, editors, *Advances in Neural Information Processing Systems 16*. MIT Press, Cambridge, MA, 2004.

[13] P. G. Martinsson and Vladimir Rokhlin. An accelerated kernel-independent fast multipole method in one dimension. *SIAM J. Scientific Computing*, 29(3):1160–1178, 2007.

[14] A. W. Moore, J. Schneider, and K. Deng. Efficient locally weighted polynomial regression predictions. In D. Fisher, editor, *Proceedings of the Fourteenth International Conference on Machine Learning*, pages 196–204, San Francisco, 1997. Morgan Kaufmann.

[15] B. W. Silverman. *Density Estimation for Statistics and Data Analysis*. Chapman and Hall/CRC, 1986.

